# Simultaneous Sampling and Multi-Structure Fitting with Adaptive Reversible Jump MCMC

**Trung Thanh Pham, Tat-Jun Chin, Jin Yu and David Suter**
School of Computer Science, The University of Adelaide, South Australia
{trung,tjchin,jin.yu,dsuter}@cs.adelaide.edu.au

## Abstract

Multi-structure model fitting has traditionally taken a two-stage approach: First, sample a (large) number of model hypotheses, then select the subset of hypotheses that optimise a joint fitting and model selection criterion. This disjoint two-stage approach is arguably suboptimal and inefficient — if the random sampling did not retrieve a good set of hypotheses, the optimised outcome will not represent a good fit. To overcome this weakness we propose a new multi-structure fitting approach based on Reversible Jump MCMC. Instrumental in raising the effectiveness of our method is an *adaptive* hypothesis generator, whose proposal distribution is learned incrementally and online. We prove that this adaptive proposal satisfies the *diminishing adaptation* property crucial for ensuring ergodicity in MCMC. Our method effectively conducts hypothesis sampling and optimisation simultaneously, and yields superior computational efficiency over previous two-stage methods.

## 1  Introduction

Multi-structure model fitting is concerned with estimating the multiple instances (or structures) of a geometric model embedded in the input data. The task manifests in applications such as mixture regression [21], motion segmentation [27, 10], and multi-projective estimation [29]. Such a problem is known for its "chicken-and-egg" nature: Both data-to-structure assignments and structure parameters are unavailable, but given the solution of one subproblem, the solution of the other can be easily derived. In practical settings the number of structures is usually unknown beforehand, thus model selection is required in conjunction to fitting. This makes the problem very challenging.

A common framework is to optimise a robust goodness-of-fit function jointly with a model selection criterion. For tractability most methods [25, 19, 17, 26, 18, 7, 31] take a "hypothesise-then-select" approach: First, randomly sample from the parameter space a large number of putative model hypotheses, then select a subset of the hypotheses (structures) that optimise the combined objective function. The hypotheses are typically fitted on minimal subsets [9] of the input data. Depending on the specific definition of the cost functions, a myriad of strategies have been proposed to select the best structures, namely tabu search [25], branch-and-bound [26], linear programming [19], dirichlet mixture clustering [17], message passing [18], graph cut [7], and quadratic programming [31].

While sampling is crucial for tractability, a disjoint two-stage approach raises an awkward situation: If the sampled hypotheses are inaccurate, or worse, if not all valid structures are sampled, the selection or optimisation step will be affected. The concern is palpable especially for higher-order geometric models (e.g., fundamental matrices in motion segmentation [27]) where enormous sampling effort is required before hitting good hypotheses (those fitted on all-inlier minimal subsets). Thus two-stage approaches are highly vulnerable to sampling inadequacies, even with theoretical assurances on the optimisation step (e.g., globally optimal over the sampled hypotheses [19, 7, 31]).

The issue above can be viewed as the lack of a stopping criterion for the sampling stage. If there is only one structure, we can easily evaluate the sample quality (e.g., consensus size) on-the-fly

and stop as soon as the prospect of obtaining a better sample becomes insignificant [9]. Under multi-structure data, it is unknown what a suitable stopping criterion is (apart from solving the overall fitting and model selection problem itself). One can consider iterative local refinement of the structures or re-sampling after data assignment [7], but the fact remains that if the initial hypotheses are inaccurate, the results of the subsequent fitting and refinement will be affected.

Clearly, an approach that *simultaneously* samples and optimises is more appropriate. To this end we propose a new method for multi-structure fitting and model selection based on Reversible Jump Markov Chain Monte Carlo (RJMCMC) [12]. By design MCMC techniques directly optimise via sampling. Despite their popular use [3] the method has not been fully explored in multi-structure fitting (a few authors have applied Monte Carlo techniques for robust estimation [28, 8], but mostly to enhance hypothesis sampling on single-structure data). We show how to exploit the reversible jump mechanism to provide a simple and effective framework for multi-structure model selection.

The bane of MCMC, however, is the difficulty in designing efficient proposal distributions. *Adaptive* MCMC techniques [4, 24] promise to alleviate this difficulty by learning the proposal distribution on-the-fly. Instrumental in raising the efficiency of our RJMCMC approach is a recently proposed hypothesis generator [6] that progressively updates the proposal distribution using generated hypotheses. Care must be taken in introducing such adaptive schemes, since a chain propagated based on a non-stationary proposal is non-Markovian, and unless the proposal satisfies certain properties [4, 24], this generally means a loss of asymptotic convergence to the target distribution.

Clearing these technical hurdles is one of our major contributions: Using emerging theory from adaptive MCMC [23, 4, 24, 11], we prove that the adaptive proposal, despite its origins in robust estimation [6], satisfies the properties required for convergence, most notably diminishing adaptation.

The rest of the paper is organised as follows: Sec. 2 formulates our goal within a clear optimisation framework, and outlines our RJMCMC approach. Sec. 3 describes the adaptive hypothesis proposal used in our method, and develops proof that it is a valid adaptive MCMC sampler. We present our experimental results in Sec. 4 and draw conclusions in Sec. 5.

## 2 Multi-Structure Fitting and Model Selection

Give input data $\mathbf{X} = \{\mathbf{x}_i\}_{i=1}^N$, usually with outliers, our goal is to recover the instances or structures $\boldsymbol{\theta}_k = \{\theta_c\}_{c=1}^k$ of a geometric model $\mathcal{M}$ embedded in $\mathbf{X}$. The number of valid structures $k$ is unknown beforehand and must also be estimated from the data. The problem domain is therefore the joint space of structure quantity and parameters $\{k, \boldsymbol{\theta}_k\}$. Such a problem is typically solved by jointly minimising fitting error and model complexity. Similar to [25, 19, 26], we use the AIC [1]

$$\{k^*, \boldsymbol{\theta}_{k^*}^*\} = \underset{\{k, \boldsymbol{\theta}_k\}}{\arg\min} -2 \log L(\boldsymbol{\theta}_k) + 2\alpha n(\boldsymbol{\theta}_k).$$

Here $L(\boldsymbol{\theta}_k)$ is the robust data likelihood and $n(\boldsymbol{\theta}_k)$ the number of parameters to define $\boldsymbol{\theta}_k$. We include a positive constant $\alpha$ to allow reweighting of the two components. Assuming i.i.d. Gaussian noise with known variance $\sigma$, the above problem is equivalent to minimising the function

$$f(k, \boldsymbol{\theta}_k) = \sum_{i=1}^N \rho\left(\frac{\min_c r_{ic}}{1.96\sigma}\right) + \alpha n(\boldsymbol{\theta}_k), \tag{1}$$

where $r_{ic} = g(\mathbf{x}_i, \theta_c)$ is the absolute residual of $\mathbf{x}_i$ to the $c$-th structure $\theta_c$ in $\boldsymbol{\theta}_k$. The residuals are subjected to a robust loss function $\rho(\cdot)$ to limit the influence of outliers; we use the biweight function [16]. Minimising a function like (1) over a vast domain $\{k, \boldsymbol{\theta}_k\}$ is a formidable task.

### 2.1 A reversible jump simulated annealing approach

Simulated annealing has proven to be effective for difficult model selection problems [2, 5]. The idea is to propagate a Markov chain for the Boltzmann distribution encapsulating (1)

$$b_T(k, \boldsymbol{\theta}_k) \propto \exp(-f(k, \boldsymbol{\theta}_k)/T) \tag{2}$$

where temperature $T$ is progressively lowered until the samples from $b_T(k, \boldsymbol{\theta}_k)$ converge to the global minima of $f(k, \boldsymbol{\theta}_k)$. Algorithm 1 shows the main body of the algorithm. Under weak regularity assumptions, there exist cooling schedules [5] that will guarantee that as $T$ tends to zero the samples from the chain will concentrate around the global minima.

To simulate $b_T(k, \boldsymbol{\theta}_k)$ we adopt a mixture of kernels MCMC approach [2]. This involves in each iteration the execution of a randomly chosen type of move to update $\{k, \boldsymbol{\theta}_k\}$. Algorithm 2 summarises the idea. We make available 3 types of moves: birth, death and local update. Birth and death moves change the number of structures $k$. These moves effectively cause the chain to jump across parameter spaces $\boldsymbol{\theta}_k$ of different dimensions. It is crucial that these trans-dimensional jumps are reversible to produce correct limiting behaviour of the chain. The following subsections explain.

---

**Algorithm 1** Simulated annealing for multi-structure fitting and model selection

---

1: Initialise temperature $T$ and state $\{k, \boldsymbol{\theta}_k\}$.
2: Simulate Markov chain for $b_T(k, \boldsymbol{\theta}_k)$ until convergence.
3: Lower temperature $T$ and repeat from Step 2 until $T \approx 0$.

---

**Algorithm 2** Reversible jump mixture of kernels MCMC to simulate $b_T(k, \boldsymbol{\theta}_k)$

---

**Require:** Last visited state $\{k, \boldsymbol{\theta}_k\}$ of previous chain, probability $\beta$ (Sec. 4 describes setting $\beta$).
1: Sample $a \sim \mathcal{U}_{[0,1]}$.
2: **if** $a \leq \beta$ **then**
3:     With probability $r_B(k)$, attempt birth move, else attempt death move.
4: **else**
5:     Attempt local update.
6: **end if**
7: Repeat from Step 1 until convergence (e.g., last $V$ moves all rejected).

---

### 2.1.1 Birth and death moves

The birth move propagates $\{k, \boldsymbol{\theta}_k\}$ to $\{k', \boldsymbol{\theta}'_{k'}\}$, with $k' = k+1$. Applying Green's [12, 22] seminal theorems on RJMCMC, the move is reversible if it is accepted with probability $\min\{1, A\}$, where

$$A = \frac{b_T(k', \boldsymbol{\theta}'_{k'})[1 - r_B(k')]/k'}{b_T(k, \boldsymbol{\theta}_k)r_B(k)q(\mathbf{u})} \left| \frac{\partial \boldsymbol{\theta}'_{k'}}{\partial(\boldsymbol{\theta}_k, \mathbf{u})} \right|. \tag{3}$$

The probability of proposing the birth move is $r_B(k)$, where $r_B(k) = 1$ for $k = 1$, $r_B(k) = 0.5$ for $k = 2, \ldots, k_{\max} - 1$, and $r_B(k_{\max}) = 0$. In other words, any move that attempts to move $k$ beyond the range $[1, k_{\max}]$ is disallowed in Step 3 of Algorithm 2. The death move is proposed with probability $1 - r_B(k)$. An existing structure is chosen randomly and deleted from $\boldsymbol{\theta}_k$. The death move is accepted with probability $\min\{1, A^{-1}\}$, with obvious changes to the notations in $A^{-1}$.

In the birth move, the extra degrees of freedom required to specify the new item in $\boldsymbol{\theta}'_{k'}$ are given by auxiliary variables $\mathbf{u}$, which are in turn proposed by $q(\mathbf{u})$. Following [18, 7, 31], we estimate parameters of the new item by fitting the geometric model $\mathcal{M}$ onto a minimal subset of the data. Thus $\mathbf{u}$ is a minimal subset of $\mathbf{X}$. The size $p$ of $\mathbf{u}$ is the minimum number of data required to instantiate $\mathcal{M}$, e.g., $p = 4$ for planar homographies, and $p = 7$ or $8$ for fundamental matrices [15]. Our approach is equivalently minimising (1) over collections $\{k, \boldsymbol{\theta}_k\}$ of minimal subsets of $\mathbf{X}$, where now $\boldsymbol{\theta}_k \equiv \{\mathbf{u}_c\}_{c=1}^k$. Taking this view the Jacobian $\partial \boldsymbol{\theta}'_{k'}/\partial(\boldsymbol{\theta}_k, \mathbf{u})$ is simply the identity matrix.

Considering only minimal subsets somewhat simplifies the problem, but there are still a colossal number of possible minimal subsets. Obtaining good overall performance thus hinges on the ability of proposal $q(\mathbf{u})$ to propose minimal subsets that are relevant, i.e., those fitted purely on inliers of valid structures in the data. One way is to learn $q(\mathbf{u})$ incrementally using generated hypotheses. We describe such a scheme [6] in Sec. 3 and prove that the adaptive proposal preserves ergodicity.

### 2.1.2 Local update

A local update does not change the model complexity $k$. The move involves randomly choosing a structure $\theta_c$ in $\boldsymbol{\theta}_k$ to update, making only local adjustments to its minimal subset $\mathbf{u}_c$. The outcome is a revised minimal subset $\mathbf{u}'_c$, and the move is accepted with probability $\min\{1, A\}$, where

$$A = \frac{b_T(k, \boldsymbol{\theta}'_k)q(\mathbf{u}_c|\theta'_c)}{b_T(k, \boldsymbol{\theta}_k)q(\mathbf{u}'_c|\theta_c)}. \tag{4}$$

As shown in the above our local update is also accomplished with the adaptive proposal $q(\mathbf{u}|\theta)$, but this time conditioned on the selected structure $\theta_c$. Sec. 3 describes and anlyses $q(\mathbf{u}|\theta)$.

# 3 Adaptive MCMC for Multi-Structure Fitting

Our work capitalises on the hypothesis generation scheme of Chin et al. called *Multi-GS* [6] originally proposed for robust geometric fitting. The algorithm maintains a series of sampling weights which are revised incrementally as new hypotheses are generated. This bears similarity to the pioneering Adaptive Metropolis (AM) method of Haario et al. [13]. Here, we prove that our adaptive proposals $q(\mathbf{u})$ and $q(\mathbf{u}|\theta)$ based on Multi-GS satisfy conditions required to preserve ergodicity.

## 3.1 The Multi-GS algorithm

Let $\{\theta_m\}_{m=1}^M$ aggregate the set of hypotheses fitted on the minimal subsets proposed thus far in all birth and local update moves in Algorithm 1. To build the sampling weights, first for each $\mathbf{x}_i \in \mathbf{X}$ we compute its absolute residuals as measured to the $M$ hypotheses, yielding the residual vector

$$\mathbf{r}^{(i)} := [\, r_1^{(i)} \; r_2^{(i)} \; \cdots \; r_M^{(i)} \,].$$

We then find the permutation

$$\mathbf{a}^{(i)} := [\, a_1^{(i)} \; a_2^{(i)} \; \cdots \; a_M^{(i)} \,]$$

that sorts the elements in $\mathbf{r}^{(i)}$ in non-descending order. The permutation $\mathbf{a}^{(i)}$ essentially ranks the $M$ hypotheses according to the preference of $\mathbf{x}_i$; The higher a hypothesis is ranked the more likely $\mathbf{x}_i$ is an inlier to it. The weight $w_{i,j}$ between the pair $\mathbf{x}_i$ and $\mathbf{x}_j$ is obtained as

$$w_{i,j} = I_h(\mathbf{x}_i, \mathbf{x}_j) := \frac{1}{h} \left| \mathbf{a}_h^{(i)} \cap \mathbf{a}_h^{(j)} \right|, \tag{5}$$

where $|\mathbf{a}_h^{(i)} \cap \mathbf{a}_h^{(j)}|$ is the number of identical elements shared by the first-$h$ elements of $\mathbf{a}^{(i)}$ and $\mathbf{a}^{(j)}$. Clearly $w_{i,j}$ is symmetric with respect to the input pair $\mathbf{x}_i$ and $\mathbf{x}_j$, and $w_{i,i} = 1$ for all $i$. To ensure technical consistency in our later proofs, we add a small positive offset $\gamma$ to the weight[1], or

$$w_{i,j} = \max(I_h(\mathbf{x}_i, \mathbf{x}_j), \gamma), \tag{6}$$

hence $\gamma \le w_{i,j} \le 1$. The weight $w_{i,j}$ measures the correlation of the top $h$ preferences of $\mathbf{x}_i$ and $\mathbf{x}_j$, and this value is typically high iff $\mathbf{x}_i$ and $\mathbf{x}_j$ are inliers from the same structure; Figs. 1 (c)–(g) illustrate. Parameter $h$ controls the discriminative power of $w_{i,j}$, and is typically set as a fixed ratio $k$ of $M$, i.e., $h = \lceil kM \rceil$. Experiments suggest that $k = 0.1$ provides generally good performance [6].

Multi-GS exploits the preference correlations to sample the next minimal subset $\mathbf{u} = \{\mathbf{x}_{s_t}\}_{t=1}^p$, where $\mathbf{x}_{s_t} \in \mathbf{X}$ and $s_t \in \{1, \ldots, N\}$ indexes the particular datum from $\mathbf{X}$; henceforth we regard $\mathbf{u} \equiv \{s_t\}_{t=1}^p$. The first datum $s_1$ is chosen purely randomly. Beginning from $t = 2$, the selection of the $t$-th member $s_t$ considers the weights related to the data $s_1, \ldots, s_{t-1}$ already present in $\mathbf{u}$. More specifically, the index $s_t$ is sampled according to the probabilities

$$P_t(i) \propto \prod_{z=1}^{t-1} w_{s_z, i}, \qquad \text{for } i = 1, \ldots, N, \tag{7}$$

i.e., if $P_t(i) > P_t(j)$ then $i$ is more likely than $j$ to be chosen as $s_t$. A new hypothesis $\theta_{M+1}$ is then fitted on $\mathbf{u}$ and the weights are updated in consideration of $\theta_{M+1}$. Experiments comparing sampling efficiency (e.g., all-inlier minimal subsets produced per unit time) show that Multi-GS is superior over previous guided sampling schemes, especially on multi-structure data; See [6] for details.

## 3.2 Is Multi-GS a valid adaptive MCMC proposal?

Our RJMCMC scheme in Algorithm 2 depends on the Multi-GS-inspired adaptive proposals $q_M(\mathbf{u})$ and $q_M(\mathbf{u}|\theta)$, where we now add the subscript $M$ to make explicit their dependency on the set of aggregated hypotheses $\{\theta_m\}_{m=1}^M$ as well as the weights $\{w_{i,j}\}_{i,j=1}^N$ they induce. The probability of proposing a minimal subset $\mathbf{u} = \{s_t\}_{t=1}^p$ from $q_M(\mathbf{u})$ can be calculated as

$$q_M(\mathbf{u}) = \frac{1}{N} \prod_{\substack{a < b \\ b \le p}} w_{s_a, s_b} \left[ \prod_{d=1}^{p-1} \mathbf{1}^T \bigodot_{e=1}^{d} \mathbf{w}_{s_e} \right]^{-1}, \tag{8}$$

where $\mathbf{w}_i$ is the column vector $[\ w_{i,1}\ \ldots\ w_{i,N}\ ]^T$ and $\odot$ is the sequential Hadamard product over the given multiplicands. The term with the inverse in Eq. (8) relates to the normalising constants for Eq. (7). As an example, the probability of selecting the minimal subset $\mathbf{u} = \{s_1, s_2, s_3, s_4\}$ is

$$q_M(\mathbf{u}) = \frac{1}{N} \frac{w_{s_1,s_2} w_{s_1,s_3} w_{s_2,s_3} w_{s_1,s_4} w_{s_2,s_4} w_{s_3,s_4}}{\mathbf{1}^T \mathbf{w}_{s_1} \mathbf{1}^T (\mathbf{w}_{s_1} \odot \mathbf{w}_{s_2}) \mathbf{1}^T (\mathbf{w}_{s_1} \odot \mathbf{w}_{s_2} \odot \mathbf{w}_{s_3})}.$$

The local update proposal $q_M(\mathbf{u}|\theta)$ differs only in the manner in which the first datum $\mathbf{x}_{s_1}$ is selected. Instead of chosen purely randomly, the first index $s_1$ is sampled according to

$$P_{s_1}(i) \propto \exp\left(-\frac{\mathcal{O}(g(\mathbf{x}_i, \theta))}{n}\right), \qquad \text{for } i = 1, \ldots, N, \tag{9}$$

where $\mathcal{O}(g(\mathbf{x}_i, \theta))$ is the order statistic of the absolute residual $g(\mathbf{x}_i, \theta)$ as measured to $\theta$; to define $q_M(\mathbf{u}|\theta)$ the $1/N$ term in Eq. (8) is simply replaced with the appropriate probability from Eq. (9). For local updates an index $i$ is more likely to be chosen as $s_1$ if $\mathbf{x}_i$ is close to $\theta$. Parameter $n$ relates to our prior belief of the minimum number of inliers per structure; we fix this to $n = 0.1N$.

Since our proposal distributions are updated with the arrival of new hypotheses, the corresponding transition probabilities are inhomogeneous (they change with time) and the chain non-Markovian (the transition to a future state depends on all previous states). We aim to show that such continual adaptations with Multi-GS will still lead to the correct target distribution (2). First we restate Theorem 1 in [11] which is distilled from other work on Adaptive MCMC [23, 4, 24].

**Theorem 1.** *Let $Z = \{Z_n : n > 0\}$ be a stochastic process on a compact state space $\Xi$ evolving according to a collection of transition kernels*

$$T_n(z, z') = pr(Z_{n+1}|Z_n = z, Z_{n-1} = z_{n-1}, \ldots, Z_0 = z_0),$$

*and let $p(z)$ be the distribution of $Z_n$. Suppose for every $n$ and $z_0, \ldots, z_{n-1} \in \Xi$ and for some distribution $\pi(z)$ on $\Xi$,*

$$\sum_{z_n} \pi(z_n) T_n(z_n, z_{n+1}) = \pi(z_{n+1}), \tag{10}$$

$$|T_{n+k}(z, z') - T_n(z, z')| \le a_n c_k, \ \ a_n = \mathcal{O}(n^{-r_1}), \ \ c_k = \mathcal{O}(k^{-r_2}), \ \ r_1, r_2 > 0, \tag{11}$$

$$T_n(z, z') \ge \epsilon \pi(z'), \ \ \epsilon > 0, \tag{12}$$

*where $\epsilon$ does not depend on $n, z_0, \ldots, z_{n-1}$. Then, for any initial distribution $p(z_0)$ for $Z_0$,*

$$\sup_{z_n} |p(z_n) - \pi(z_n)| \to 0 \ \text{for} \ n \to \infty.$$

**Diminishing adaptation.** Eq. (11) dictates that the transition kernel, and thus the proposal distribution in the Metropolis-Hastings updates in Eqs. (3) and (4), must converge to a fixed distribution, i.e., the adaptation must diminish. To see that this occurs naturally in $q_M(\mathbf{u})$, first we show that $w_{i,j}$ for all $i, j$ converges as $M$ increases. Without loss of generality assume that $b$ new hypotheses are generated between successive weight updates $w_{i,j}$ and $w'_{i,j}$. Then,

$$\lim_{M \to \infty} \left|w'_{i,j} - w_{i,j}\right| = \lim_{M \to \infty} \left|\frac{|\mathbf{a}'^{(i)}_{k(M+b)} \cap \mathbf{a}'^{(j)}_{k(M+b)}|}{k(M+b)} - \frac{|\mathbf{a}^{(i)}_{kM} \cap \mathbf{a}^{(j)}_{kM}|}{kM}\right| \le \lim_{M \to \infty} \left|\frac{|\mathbf{a}^{(i)}_{kM} \cap \mathbf{a}^{(j)}_{kM}| \pm b(k+1)}{k(M+b)} - \frac{|\mathbf{a}^{(i)}_{kM} \cap \mathbf{a}^{(j)}_{kM}|}{kM}\right|$$

$$= \lim_{M \to \infty} \left|\frac{|\mathbf{a}^{(i)}_{kM} \cap \mathbf{a}^{(j)}_{kM}|/M \pm b(k+1)/M}{k + kb/M} - \frac{|\mathbf{a}^{(i)}_{kM} \cap \mathbf{a}^{(j)}_{kM}|/M}{k}\right| = 0,$$

where $\mathbf{a}'^{(i)}$ is the revised preference of $\mathbf{x}_i$ in consideration of the $b$ new hypotheses. The result is based on the fact that the extension of $b$ hypotheses will only perturb the overlap between the top-$k$ percentile of any two preference vectors by at most $b(k+1)$ items. It should also be noted that the result is not due to $w'_{i,j}$ and $w_{i,j}$ simultaneously vanishing with increasing $M$; in general

$$\lim_{M \to \infty} |\mathbf{a}^{(i)}_{kM} \cap \mathbf{a}^{(j)}_{kM}|/M \ne 0$$

since $\mathbf{a}^{(i)}$ and $\mathbf{a}^{(j)}$ are extended and revised as $M$ increases and this may increase their mutual overlap. Figs. 1 (c)–(g) illustrate the convergence of $w_{i,j}$ as $M$ increases. Using the above result, it can be shown that the product of any two weights also converges

$$\lim_{M \to \infty} \left|w'_{i,j} w'_{p,q} - w_{i,j} w_{p,q}\right| = \lim_{M \to \infty} \left|w'_{i,j}(w'_{p,q} - w_{p,q}) + w_{p,q}(w'_{i,j} - w_{i,j})\right|$$

$$\le \lim_{M \to \infty} \left|w'_{i,j}\right| \left|w'_{p,q} - w_{p,q}\right| + \left|w_{p,q}\right| \left|w'_{i,j} - w_{i,j}\right| = 0.$$

This result is readily extended to the product of any number of weights. To show the convergence of the normalisation terms in (8), we first observe that the sum of weights is bounded away from 0

$$\forall i, \qquad \mathbf{1}^T \mathbf{w}_i \geq \mathcal{L}, \qquad \mathcal{L} > 0,$$

due to the offsetting (6) and the constant element $w_{i,i} = 1$ in $\mathbf{w}_i$ (although $w_{i,i}$ will be set to zero to enforce sampling without replacement [6]). It can thus be established that

$$\lim_{M \to \infty} \left| \frac{1}{\mathbf{1}^T \mathbf{w}_i'} - \frac{1}{\mathbf{1}^T \mathbf{w}_i} \right| = \lim_{M \to \infty} \left| \frac{\mathbf{1}^T \mathbf{w}_i' - \mathbf{1}^T \mathbf{w}_i}{(\mathbf{1}^T \mathbf{w}_i')(\mathbf{1}^T \mathbf{w}_i)} \right| \leq \lim_{M \to \infty} \left| \frac{\mathbf{1}^T \mathbf{w}_i' - \mathbf{1}^T \mathbf{w}_i}{\mathcal{L}^2} \right| = 0$$

since the sum of the weights also converges. The result is readily extended to the inverse of the sum of any number of Hadamard products of weights, since we have also previously established that the product of any number of weights converges. Finally, since Eq. (8) involves only multiplications of convergent quantities, $q_M(\mathbf{u})$ will converge to a fixed distribution as the update progresses.

**Invariance.** Eq. (10) requires that transition probabilities based on $q_M(\mathbf{u})$ permits an invariant distribution individually for all $M$. Since we propose and accept based on the Metropolis-Hastings algorithm, detailed balance is satisfied by construction [3], which means that a Markov chain propagated based on $q_M(\mathbf{u})$ will asymptotically sample from the target distribution.

**Uniform ergodicity.** Eq. (12) requires that $q_M(\mathbf{u})$ for all $M$ be individually ergodic, i.e., the resulting chain using $q_M(\mathbf{u})$ is aperiodic and irreducible. Again, since we simulate the target using Metropolis-Hastings, every proposal has a chance of being rejected, thus implying aperiodicity [3]. Irreducibility is satisfied by the offsetting in (6) and renormalising [20], since this implies that there is always a non-zero probability of reaching any state (minimal subset) from the current state.

The above results apply for the local update proposal $q_M(\mathbf{u}|\theta)$ which differs from $q_M(\mathbf{u})$ only in the (stationary) probability to select the first index $s_1$. Hence $q_M(\mathbf{u}|\theta)$ is also a valid adaptive proposal.

## 4 Experiments

We compare our approach (ARJMC) against state-of-the-art methods: message passing [18] (FLOSS), energy minimisation with graph cut [7] (ENERGY), and quadratic programming based on a novel preference feature [31] (QP-MF). We exclude older methods with known weaknesses, e.g., computational inefficiency [19, 17, 26], low accuracy due to greedy search [25], or vulnerability to outliers [17]. All methods are run in MATLAB except ENERGY which is available in C++[2].

For ARJMC, standard deviation $\sigma$ in (1) is set as $t/1.96$, where $t$ is the inlier threshold [9] obtained using ground truth model fitting results— The same $t$ is provided to the competitors. In Algorithm 1 temperature $T$ is initialiased as 1 and we apply the geometric cooling schedule $T_{next} = 0.99T$. In Algorithm 2, probability $\beta$ is set as equal to current temperature $T$, thus allowing more global explorations in the parameter space initially before concentrating on local refinement subsequently. Such a helpful strategy is not naturally practicable in disjoint two-stage approaches.

### 4.1 Two-view motion segmentation

The goal is to segment point trajectories $\mathbf{X}$ matched across two views into distinct motions [27]. Trajectories of a particular motion can be related by a distinct fundamental matrix $\mathbf{F} \in \mathbb{R}^{3 \times 3}$ [15]. Our task is thus to estimate the number of motions $k$ and the fundamental matrices $\{\mathbf{F}_c\}_{c=1}^k$ corresponding to the motions embedded in data $\mathbf{X}$. Note that $\mathbf{X}$ may contain false trajectories (outliers). We estimate fundamental matrix hypotheses from minimal subsets of size $p = 8$ using the 8-point method [14]. The residual $g(\mathbf{x}_i, \mathbf{F})$ is computed as the Sampson distance [15].

We test the methods on publicly available two-view motion segmentation datasets [30]. In particular we test on the 3- and 4-motion datasets provided, namely *breadtoycar*, *carchipscube*, *toycubecar*, *breadcubechips*, *biscuitbookbox*, *cubebreadtoychips* and *breadcartoychips*; see the dataset homepage for more details. Correspondences were established via SIFT matching and manual filtering was done to obtain ground truth segmentation. Examples are shown in Figs. 1(a) and 1(b).

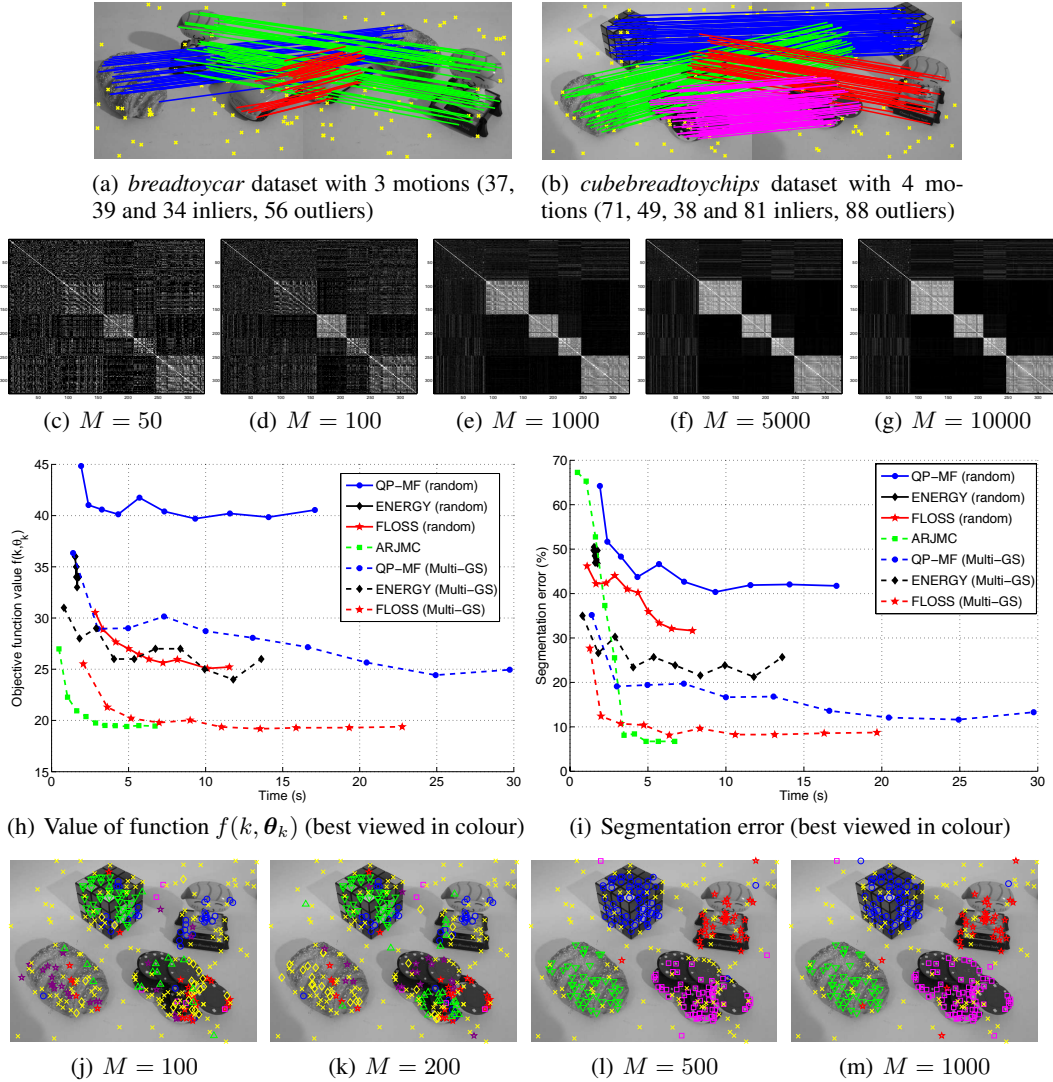

(a) *breadtoycar* dataset with 3 motions (37, 39 and 34 inliers, 56 outliers)

(b) *cubebreadtoychips* dataset with 4 motions (71, 49, 38 and 81 inliers, 88 outliers)

(c) $M = 50$   (d) $M = 100$   (e) $M = 1000$   (f) $M = 5000$   (g) $M = 10000$

(h) Value of function $f(k, \boldsymbol{\theta}_k)$ (best viewed in colour)

(i) Segmentation error (best viewed in colour)

(j) $M = 100$   (k) $M = 200$   (l) $M = 500$   (m) $M = 1000$

Figure 1: (a) and (b) show respectively a 3- and 4-motion dataset (colours show ground truth labelling). To minimise clutter, lines joining false matches are not drawn. (c)–(g) show the evolution of the matrix of pairwise weights (5) computed from (b) as the number of hypotheses $M$ is increased. For presentation the data are arranged according to their structure membership, which gives rise to a 4-block pattern. Observe that the block pattern, hence weights, converge as $M$ increases. (h) and (i) respectively show performance measures (see text) of four methods on the dataset in (b). (j)–(m) show the evolution of the labelling result of ARJMC as $M$ increases (only one view is shown).

Figs. 1(c)–(g) show the evolution of the pairwise weights (5) as $M$ increases until 10,000 for the data in Fig. 1(b). The matrices exhibit a a four-block pattern, indicating strong mutual preference among inliers from the same structure. This phenomenon allows accurate selection of minimal subsets in Multi-GS [6]. More pertinently, as we predicted in Sec. 3.2, the weights converge as $M$ increases, as evidenced by the stabilising block pattern. Note that only a small number of weights are actually computed in Multi-GS [6]; the full matrix of weights are calculated here for illustration only.

We run ARJMC and record the following performance measures: Value of the objective function $f(k, \boldsymbol{\theta}_k)$ in Eq. (1), and segmentation error. The latter involves assigning each datum $\mathbf{x}_i \in \mathbf{X}$ to the nearest structure in $\boldsymbol{\theta}_k$ *if* the residual is less than the threshold $t$; else $\mathbf{x}_i$ is labelled as an outlier. The overall labelling error is then obtained. The measures are recorded at *time intervals* corresponding to the instances when $M = 100, 200, \ldots, 1000$ number of hypotheses generated so far in Algorithm 1. Median results over 20 repetitions on the data in Fig. 1(b) are shown in Figs. 1(h) and 1(i). Figs. 1(j)–1(m) depict the evolution of the segmentation result of ARJMC as $M$ increases.

For objective comparisons the competing *two-stage* methods were tested as follows: First, $M = 100, 200, \ldots, 1000$ hypotheses are accumulatively generated (using both uniform random sampling [9] and Multi-GS [6]). A *new* instance of each method is invoked on each set of $M$ hypotheses. We ensure that each method returns the true number of structures *for all* $M$; this represents an advantage over ARJMC, since the "online learning" nature of ARJMC means the number of structures is not discovered until closer to convergence. Results are also shown in Figs. 1(h) and 1(i).

Firstly, it is clear that the performance of the two-stage methods on both measures are improved dramatically with the application of Multi-GS for hypothesis generation. From Fig. 1(h) ARJMC is the most efficient in minimising the function $f(k, \boldsymbol{\theta}_k)$; it converges to a low value in significantly less time. It should be noted however that the other methods are not directly minimising AIC or $f(k, \boldsymbol{\theta}_k)$. The segmentation error (which no method here is directly minimising) thus represents a more objective performance measure. From Fig. 1(i), it can be seen that the initial error of ARJMC is much higher than all other methods, a direct consequence of not having yet estimated the true number of structures. The error is eventually minimised as ARJMC converges. Table 1 which summarises the results on the other datasets (all using Multi-GS) conveys a similar picture. Further results on multi-homography detection also yield similar outcomes (see supplementary material).

| Dataset | *breadtoycar* (3 structures) | | | | *carchipscube* (3 structures) | | | | *toycubecar* (3 structures) | | | |
| # inliers, outliers | 37, 39 and 34 inliers, 56 outliers | | | | 19, 33 and 53 inliers, 60 outliers | | | | 45, 69 and 14 inliers, 72 outliers | | | |
| M | FLOSS | ENERGY | QP-MF | ARJMC | FLOSS | ENERGY | QP-MF | ARJMC | FLOSS | ENERGY | QP-MF | ARJMC |
|---|---|---|---|---|---|---|---|---|---|---|---|---|
| 100 | 25.22 | 31.74 | 24.78 | 68.70 | 21.82 | 29.70 | 23.64 | 52.73 | 31.75 | 26.25 | 29.00 | 81.50 |
| 200 | 14.13 | 26.74 | 18.91 | 61.96 | 15.76 | 36.97 | 30.30 | 58.18 | 23.00 | 27.25 | 19.25 | 75.75 |
| 300 | 10.43 | 33.48 | 18.70 | 54.13 | 12.73 | 24.24 | 26.67 | 49.09 | 22.75 | 25.25 | 18.00 | 65.00 |
| 400 | 9.57 | 27.83 | 18.26 | 48.48 | 10.30 | 32.73 | 28.48 | 24.24 | 22.00 | 26.25 | 22.50 | 52.75 |
| 500 | 9.57 | 27.39 | 26.30 | 10.87 | 10.30 | 30.91 | 27.27 | 13.33 | 22.50 | 22.50 | 23.00 | 45.75 |
| 600 | 8.70 | 25.87 | 20.43 | 8.48 | 9.09 | 28.48 | 23.03 | 9.70 | 21.75 | 26.50 | 20.75 | 37.75 |
| 700 | 8.91 | 30.43 | 21.30 | 7.17 | **8.48** | 22.42 | 27.88 | 9.70 | 17.50 | 26.50 | 23.00 | 23.50 |
| 800 | 7.83 | 21.09 | 22.17 | **6.52** | 10.30 | 26.67 | 25.45 | 9.70 | 21.50 | 26.50 | 20.00 | 18.50 |
| 900 | 7.39 | 25.22 | 26.74 | 6.52 | 8.48 | 36.36 | 26.06 | 9.70 | 18.75 | 20.75 | 15.75 | 19.75 |
| 1000 | 7.17 | 20.43 | 25.22 | 6.52 | 9.09 | 28.48 | 23.64 | 9.70 | **15.50** | 23.00 | 18.25 | 19.50 |
| Time (seconds) | 12.88 | 9.40 | 21.57 | **5.44** | 9.57 | 7.02 | 16.23 | **5.16** | 11.73 | 8.14 | 18.94 | **4.95** |
| Dataset | *breadcubechip* (3 structures) | | | | *breadcartoychip* (4 structures) | | | | *biscuitbookbox* (3 structures) | | | |
| # inliers, outliers | 34, 57 and 58 inliers, 81 outliers | | | | 33, 23, 41 and 58 inliers, 82 outliers | | | | 67, 41 and 54 inliers, 97 outliers | | | |
| M | FLOSS | ENERGY | QP-MF | ARJMC | FLOSS | ENERGY | QP-MF | ARJMC | FLOSS | ENERGY | QP-MF | ARJMC |
| 100 | 23.49 | 21.08 | 24.10 | 81.93 | 36.92 | 35.86 | 32.07 | 54.01 | 17.57 | 25.87 | 18.15 | 49.03 |
| 200 | 16.27 | 13.25 | 15.06 | 78.92 | 28.90 | 27.00 | 20.04 | 61.60 | 11.00 | 17.95 | 17.76 | 31.85 |
| 300 | 12.65 | 10.84 | 18.07 | 70.48 | 19.41 | 21.30 | 17.09 | 61.18 | 7.92 | 17.95 | 9.27 | 6.95 |
| 400 | 13.86 | 11.45 | 14.46 | 48.80 | 17.51 | 20.88 | 15.19 | 56.54 | 8.49 | 14.86 | 13.51 | 6.37 |
| 500 | 12.05 | 13.25 | 13.25 | 37.95 | 13.92 | 18.56 | 13.50 | 21.94 | 7.92 | 18.73 | 10.04 | **4.44** |
| 600 | 12.05 | 12.05 | 12.05 | 11.45 | 11.81 | 19.83 | 13.92 | 18.99 | 5.79 | 17.18 | 11.39 | 5.21 |
| 700 | 10.84 | 11.45 | 9.04 | 9.64 | 10.76 | 15.18 | 12.66 | 18.14 | 5.79 | 18.92 | 14.67 | 4.83 |
| 800 | 10.84 | 12.05 | 11.45 | 9.64 | 10.55 | 18.56 | 12.24 | 10.97 | 5.79 | 16.60 | 13.51 | 5.21 |
| 900 | 10.84 | 10.24 | 10.24 | **7.83** | 10.34 | 14.55 | 11.39 | **9.70** | 5.79 | 18.53 | 12.36 | 5.21 |
| 1000 | 10.84 | 10.84 | 10.84 | 8.43 | **9.70** | 15.18 | 11.60 | 9.70 | 5.79 | 13.71 | 13.13 | 5.79 |
| Time (seconds) | 9.57 | 6.96 | 16.38 | **4.47** | 13.40 | 9.86 | 22.46 | **5.39** | 15.46 | 10.66 | 24.36 | **5.47** |

Table 1: Median segmentation error (%) at different number of hypotheses $M$. Time elapsed at $M = 1000$ is shown at the bottom. The lowest error and time achieved on each dataset is boldfaced.

# 5 Conclusions

By design, since our algorithm conducts hypothesis sampling, geometric fitting and model selection simultaneously, it minimises wastage in the sampling process and converges faster than previous two-stage approaches. This is evident from the experimental results. Underpinning our novel Reversible Jump MCMC method is an efficient hypothesis generator whose proposal distribution is learned online. Drawing from new theory on Adaptive MCMC, we prove that our efficient hypothesis generator satisfies the properties crucial to ensure convergence to the correct target distribution. Our work thus links the latest developments from MCMC optimisation and geometric model fitting.

**Acknowledgements.** The authors would like to thank Anders Eriksson his insightful comments. This work was partly supported by the Australian Research Council grant DP0878801.

## Footnotes

[1] It can be shown if both $\mathbf{x}_i$ and $\mathbf{x}_j$ are uniformly distributed outliers, the expected value of $w_{i,j}$ is $h/M$, i.e., a given pair $\mathbf{x}_i$ and $\mathbf{x}_j$ will likely have non-zero preference correlation.

[2]http://vision.csd.uwo.ca/code/#Multi-label_optimization

# References

[1] H. Akaike. A new look at the statistical model identification. *IEEE Trans. on Automatic Control*, 19(6):716–723, 1974.

[2] C. Andrieu, N. de Freitas, and A. Doucet. Robust full Bayesian learning for radial basis networks. *Neural Computation*, 13:2359–2407, 2001.

[3] C. Andrieu, N. de Freitas, A. Doucet, and M. I. Jordan. An introduction to MCMC for machine learning. *Machine Learning*, 50:5–43, 2003.

[4] C. Andrieu and J. Thoms. A tutorial on adaptive MCMC. *Statistics and Computing*, 18(4), 2008.

[5] S. P. Brooks, N. Friel, and R. King. Classical model selection via simulated annealing. *J. R. Statist. Soc. B*, 65(2):503–520, 2003.

[6] T.-J. Chin, J. Yu, and D. Suter. Accelerated hypothesis generation for multi-structure robust fitting. In *European Conf. on Computer Vision*, 2010.

[7] A. Delong, A. Osokin, H. Isack, and Y. Boykov. Fast approximate energy minimization with label costs. In *Computer Vision and Pattern Recognition*, 2010.

[8] L. Fan and T. Pylnänäinen. Adaptive sample consensus for efficient random optimisation. In *Int. Symposium on Visual Computing*, 2009.

[9] M. A. Fischler and R. C. Bolles. Random sample consensus: A paradigm for model fitting with applications to image analysis and automated cartography. *Comm. of the ACM*, 24:381–395, 1981.

[10] S. Gaffney and P. Smyth. Trajectory clustering with mixtures of regression models. In *ACM SIG on Knowledge Discovery and Data Mining*, 1999.

[11] P. Giordani and R. Kohn. Adaptive independent Metropolis-Hastings by fast estimation of mixtures of normals. *Journal of Computational and Graphical Statistics*, 19(2):243–259, 2010.

[12] P. J. Green. Reversible jump Markov chain Monte Carlo computation and Bayesian model determination. *Biometrika*, 82(4):711–732, 1995.

[13] H. Haario, E. Saksman, and J. Tamminen. An adaptive Metropolis algorithm. *Bernoulli*, 7(2):223–242, 2001.

[14] R. Hartley. In defense of the eight-point algorithm. *IEEE Trans. on Pattern Analysis and Machine Intelligence*, 19(6):580–593, 1997.

[15] R. Hartley and A. Zisserman. *Multiple View Geometry*. Cambridge University Press, 2004.

[16] P. J. Huber. *Robust Statistics*. John Wiley & Sons Inc., 2009.

[17] Y.-D. Jian and C.-S. Chen. Two-view motion segmentation by mixtures of dirichlet process with model selection and outlier removal. In *International Conference on Computer Vision*, 2007.

[18] N. Lazic, I. Givoni, B. Frey, and P. Aarabi. FLoSS: Facility location for subspace segmentation. In *IEEE Int. Conf. on Computer Vision*, 2009.

[19] H. Li. Two-view motion segmentation from linear programming relaxation. In *Computer Vision and Pattern Recognition*, 2007.

[20] D. Nott and R. Kohn. Adaptive sampling for Bayesian variable selection. *Biometrika*, 92:747–763, 2005.

[21] N. Quadrianto, T. S. Caetano, J. Lim, and D. Schuurmans. Convex relaxation of mixture regression with efficient algorithms. In *Advances in Neural Information Processing Systems*, 2010.

[22] S. Richardson and P. J. Green. On Bayesian analysis on mixtures with an unknown number of components. *J. R. Statist. Soc. B*, 59(4):731–792, 1997.

[23] G. O. Roberts and J. S. Rosenthal. Coupling and ergodicity of adaptive Markov chain Monte Carlo algorithms. *Journal of Applied Probability*, 44:458–475, 2007.

[24] G. O. Roberts and J. S. Rosenthal. Examples of adaptive MCMC. *Journal of Computational and Graphical Statistics*, 18(2):349–367, 2009.

[25] K. Schinder and D. Suter. Two-view multibody structure-and-motion with outliers through model selection. *IEEE Trans. on Pattern Analysis and Machine Intelligence*, 28(6):983–995, 2006.

[26] N. Thakoor and J. Gao. Branch-and-bound hypothesis selection for two-view multiple structure and motion segmentation. In *Computer Vision and Pattern Recognition*, 2008.

[27] P. H. S. Torr. *Motion segmentation and outlier detection*. PhD thesis, Dept. of Engineering Science, University of Oxford, 1995.

[28] P. H. S. Torr and C. H. Davidson. IMPSAC: Synthesis of importance sampling and random sample consensus. *IEEE Trans. on Pattern Analysis and Machine Intelligence*, 25(3):354–364, 2003.

[29] E. Vincent and R. Laganière. Detecting planar homographies in an image pair. In *International Symposium on Image and Signal Processing and Analysis*, 2001.

[30] H. S. Wong, T.-J. Chin, J. Yu, and D. Suter. Dynamic and hierarchical multi-structure geometric model fitting. In *International Conference on Computer Vision*, 2011.

[31] J. Yu, T.-J. Chin, and D. Suter. A global optimization approach to robust multi-model fitting. In *Computer Vision and Pattern Recognition*, 2011.

